# Kernel Projection Machine: a New Tool for Pattern Recognition[*]

**Gilles Blanchard**
Fraunhofer First (IDA),
Kékuléstr. 7, D-12489 Berlin, Germany
`blanchar@first.fhg.de`

**Pascal Massart**
Département de Mathématiques,
Université Paris-Sud,
Bat. 425, F-91405 Orsay, France
`Pascal.Massart@math.u-psud.fr`

**Régis Vert**
LRI, Université Paris-Sud,
Bat. 490, F-91405 Orsay, France
Masagroup
24 Bd de l'Hopital, F-75005 Paris, France
`Regis.Vert@lri.fr`

**Laurent Zwald**
Département de Mathématiques,
Université Paris-Sud,
Bat. 425, F-91405 Orsay, France
`Laurent.Zwald@math.u-psud.fr`

## Abstract

This paper investigates the effect of Kernel Principal Component Analysis (KPCA) within the classification framework, essentially the regularization properties of this dimensionality reduction method. KPCA has been previously used as a pre-processing step before applying an SVM but we point out that this method is somewhat redundant from a regularization point of view and we propose a new algorithm called *Kernel Projection Machine* to avoid this redundancy, based on an analogy with the statistical framework of regression for a Gaussian white noise model. Preliminary experimental results show that this algorithm reaches the same performances as an SVM.

## 1 Introduction

Let $(x_i, y_i)_{i=1\ldots n}$ be $n$ given realizations of a random variable $(X, Y)$ living in $\mathcal{X} \times \{-1; 1\}$. Let $P$ denote the marginal distribution of $X$. The $x_i$'s are often referred to as *inputs* (or *patterns*), and the $y_i$'s as *labels*. Pattern recognition is concerned with finding a *classifier*, i.e. a function that assigns a label to any new input $x \in \mathcal{X}$ and that makes as few prediction errors as possible.

It is often the case with real world data that the dimension of the patterns is very large, and some of the components carry more noise than information. In such cases, reducing the dimension of the data before running a classification algorithm on it sounds reasonable. One of the most famous methods for this kind of pre-processing is PCA, and its kernelized version (KPCA), introduced in the pioneering work of Schölkopf, Smola and Müller [8].

[*]This work was supported in part by the IST Programme of the European Community, under the PASCAL Network of Excellence, IST-2002-506778.

Now, whether the quality of a given classification algorithm can be significantly improved by using such pre-processed data still remains an open question. Some experiments have already been carried out to investigate the use of KPCA for classification purposes, and numerical results are reported in [8]. The authors considered the USPS handwritten digit database and reported the test error rates achieved by the linear SVM trained on the data pre-processed with KPCA: the conclusion was that the larger the number of principal components, the better the performance. In other words, the KPCA step was useless or even counterproductive.

This conclusion might be explained by a redundancy arising in their experiments: there is actually a double regularization, the first corresponding to the dimensionality reduction achieved by KPCA, and the other to the regularization achieved by the SVM. With that in mind it does not seem so surprising that KPCA does not help in that case: whatever the dimensionality reduction, the SVM anyway achieves a (possibly strong) regularization.

Still, de-noising the data using KPCA seems relevant. The aforementioned experiments suggest that KPCA should be used together with a classification algorithm that is not regularized (e.g. a simple empirical risk minimizer): in that case, it should be expected that the KPCA is by itself sufficient to achieve regularization, the choice of the dimension being guided by adequate model selection.

In this paper, we propose a new algorithm, called the Kernel Projection Machine (KPM), that implements this idea: an optimal dimension is sought so as to minimize the test error of the resulting classifier. A nice property is that the training labels are used to select the optimal dimension – optimal means that the resulting $D$-dimensional representation of the data contains the right amount of information needed to classify the inputs. To sum up, the KPM can be seen as a dimensionality-reduction-based classification method that takes into account the labels for the dimensionality reduction step.

This paper is organized as follows: Section 2 gives some statistical background on regularized method vs. projection methods. Its goal is to explain the motivation and the "Gaussian intuition" that lies behind the KPM algorithm from a statistical point of view. Section 3 explicitly gives the details of the algorithm; experiments and results, which should be considered preliminary, are reported in Section 4.

## 2 Motivations for the Kernel Projection Machine

### 2.1 The Gaussian Intuition: a Statistician's Perspective

Regularization methods have been used for quite a long time in non parametric statistics since the pioneering works of Grace Wahba in the eighties (see [10] for a review). Even if the classification context has its own specificity and offers new challenges (especially when the explanatory variables live in a high dimensional Euclidean space), it is good to remember what is the essence of regularization in the simplest non parametric statistical framework: the Gaussian white noise.

So let us assume that one observes a noisy signal $dY(x) = s(x)dx + \frac{1}{\sqrt{n}}dw(x)$, $Y(0) = 0$ on [0,1] where $dw(x)$ denotes standard white noise. To the reader not familiar with this model, it should be considered as nothing more but an idealization of the well-known fixed design regression problem $Y_i = s(i/n) + \varepsilon_i$ for $i = 1, \ldots, n$, where $\varepsilon_i \sim N(0, 1)$, where the goal is to recover the regression function $s$. (The white noise model is actually simpler to study from a mathematical point of view). The least square criterion is defined as

$$\gamma_n(f) = \|f\|^2 - 2 \int_0^1 f(x)dY(x)$$

for every $f \in L_2([0, 1])$.

Given a Mercer kernel $k$ on $[0, 1] \times [0, 1]$, the regularization least square procedure proposes

to minimize

$$\gamma_n(f) + \zeta_n \|f\|_{\mathcal{H}_k} \tag{1}$$

where $(\zeta_n)$ is a conveniently chosen sequence and $\mathcal{H}_k$ denotes the RKHS induced by $k$. This procedure can indeed be viewed as a model selection procedure since minimizing $\gamma_n(f) + \zeta_n \|f\|_{\mathcal{H}_k}$ amounts to minimizing

$$\inf_{\|f\| \leq R} \left[ \gamma_n(f) + \zeta_n R^2 \right]$$

over $R > 0$. In other words, regularization aims at selecting the "best" RKHS ball $\{f, \|f\| \leq R\}$ to represent our data.

At this stage, it is interesting to realize that the balls in the RKHS space can be viewed as ellipsoids in the original Hilbert space $L_2([0,1])$. Indeed, let $(\phi_i)_{i=1}^\infty$ be some orthonormal basis of eigenfunctions for the compact and self adjoint operator

$$T_k : f \longrightarrow \int_0^1 k(x,y)f(x)dx$$

Then, setting $\beta_j = \int_0^1 f(x)\phi_j(x)dx$ one has $\|f\|_{\mathcal{H}_k}^2 = \sum_{j=1}^\infty \frac{\beta_j^2}{\lambda_j}$ where $(\lambda_j)_{j\geq 1}$ denotes the non increasing sequence of eigenvalues corresponding to $(\phi_j)_{j\geq 1}$. Hence

$$\{f, \|f\|_{\mathcal{H}_k} \leq R\} = \left\{ \sum_{j=1}^\infty \beta_j \phi_j \; ; \; \sum_{j=1}^\infty \frac{\beta_j^2}{\lambda_j} \leq R^2 \right\}.$$

Now, due to the approximation properties of the finite dimensional spaces $\{\phi_j, \ j \leq D\}$, $D \in \mathbb{N}^*$ with respect to the ellipsoids, one can think of penalized finite dimensional projection as an alternative method to regularization. More precisely, if $\widehat{s}_D$ denotes the projection estimator on $\langle \phi_j, \ j \leq D \rangle$, i.e. $\widehat{s}_D = \sum_{j=1}^D \left( \int \phi_j dY \right) \phi_j$ and one considers the penalized selection criterion $\widehat{D} = \operatorname*{argmin}_D [\gamma_n(\widehat{s}_D) + \frac{2D}{n}]$ then, it is proved in [1] that the selected estimator $\widehat{s}_{\widehat{D}}$ obeys to the following oracle inequality

$$\mathbb{E}[\|s - \widehat{s}_{\widehat{D}}\|^2] \leq C \inf_{D \geq 1} \left[ \mathbb{E}\|s - \widehat{s}_D\|^2 \right]$$

where $C$ is some absolute constant.

The nice thing is that whenever $s$ belongs to some ellipsoid

$$\mathcal{E}(c) = \left\{ \sum_{j=1}^\infty \beta_j \phi_j \; : \; \sum_{j=1}^\infty \frac{\beta_j^2}{c_j^2} \leq 1 \right\}$$

where $(c_j)_{j\geq 1}$ is a decreasing sequence tending to $0$ as $j \to \infty$, then

$$\inf_{D \geq 1} \mathbb{E}\left[ \|s - \widehat{s}_D\|^2 \right] = \inf_{D \geq 1} \left[ \inf_{t \in S_D} \|s - t\|^2 + \frac{D}{n} \right] \leq \inf_{D \geq 1} \left[ c_D^2 + \frac{D}{n} \right]$$

As shown in [5] $\inf_{D \geq 1}[c_D^2 + \frac{D}{n}]$ is (up to some absolute constant) of the order of magnitude of the minimax risk over $\mathcal{E}(c)$.

As a consequence, the estimator $\widehat{s}_{\widehat{D}}$ is simultaneously minimax over the collection of *all* ellipsoids $\mathcal{E}(c)$, which in particular includes the collection $\{\mathcal{E}(\sqrt{\lambda}R), \ R > 0\}$.

To conclude and summarize, from a statistical performance point of view, what we can expect from a regularized estimator $\widehat{s}$ (i.e. a minimizer of (1)) is that a convenient device of $\zeta_n$ ensures that $\widehat{s}$ is simultaneously minimax over the collection of ellipsoids $\{\mathcal{E}(\sqrt{\lambda}R), \ R > 0\}$, (at least as far as asymptotic rates of convergence are concerned ). The alternative estimator $\widehat{s}_{\widehat{D}}$ actually achieves this goal and even better since it is also adaptive over the collection of all ellipsoids and not only the family $\{\mathcal{E}(\sqrt{\lambda}R), \ R > 0\}$.

## 2.2 Extension to a general classification framework

In this section we go back to classification framework as described in the introduction. First of all, it has been noted by several authors ([6],[9]) that the SVM can be seen as a regularized estimation method, where the regularizer is the squared norm of the function in $\mathcal{H}_k$. Precisely, the SVM algorithm solves the following unconstrained optimization problem:

$$\min_{f \in \mathcal{H}_k^b} \frac{1}{n} \sum_{i=1}^{n} (1 - y_i f(x_i))_+ + \lambda \|f\|_{\mathcal{H}_k}^2 \,, \tag{2}$$

where $\mathcal{H}_k^b = \{f(x) + b, \ f \in \mathcal{H}_k, b \in \mathbb{R}\}$.

The above regularization can be viewed as a model selection process over RKHS balls, similarly to the previous section. Now, the line of ideas developed there suggests that it might actually be a better idea to consider a sequence of finite-dimensional estimators. Additionally, it has been shown in [4] that the regularization term of the SVM is actually too strong. We therefore transpose the ideas of previous Gaussian case to the classification framework. Consider a Mercer kernel $k$ defined on $\mathcal{X} \times \mathcal{X}$ and Let $T_k$ denote the operator associated with kernel $k$ in the following way

$$T_k \ : \ f(.) \in L_2(\mathcal{X}) \longmapsto \int_{\mathcal{X}} k(x,.)f(x)dP(x) \in L_2(\mathcal{X})$$

Let $\phi_1, \phi_2, \ldots$ denote the eigenvectors of $T_k$, ordered by decreasing associated eigenvalues $(\lambda_i)_{1 \geq 1}$. For each integer $D$, the subspace $\mathcal{F}_D$ defined by $\mathcal{F}_D = \text{span}\{\mathbb{1}, \phi_1, \ldots, \phi_D\}$ (where $\mathbb{1}$ denotes the constant function equal to 1) corresponds to a subspace of $\mathcal{H}_k^b$ associated with kernel $k$, and $\mathcal{H}_k^b = \overline{\bigcup_{D=1}^{\infty} \mathcal{F}_D}$. Instead of selecting the "best" ball in the RKHS, as the SVM does, we consider the analogue of the projection estimator $\widehat{s}_D$:

$$\hat{f}_D = \arg \min_{f \in \mathcal{F}_D} \sum_{i=1}^{n} (1 - y_i f(x_i))_+ \tag{3}$$

that is, more explicitly,

$$\hat{f}_D(.) = \sum_{j=1}^{D} \beta_j^* \phi_j(.) + b^*$$

with

$$(\beta^*, b^*) = \arg \min_{(\beta \in \mathbb{R}^D, b \in \mathbb{R})} \sum_{i=1}^{n} \left(1 - y_i \left(\sum_{j=1}^{D} \beta_j \phi_j(x_i) + b\right)\right)_+ \tag{4}$$

An appropriate $D$ can then be chosen using an adequate model selection procedure such as penalization; we do not address this point in detail in the present work but it is of course the next step to be taken.

Unfortunately, since the underlying probability $P$ is unknown, neither are the eigenfunctions $\phi_1, \ldots$, and it is therefore not possible to implement this procedure directly. We thus resort to considering empirical quantities as will be explained in more detail in section 3. Essentially, the unknown vectorial space spanned by the first eigenfunctions of $T_k$ is replaced by the space spanned by the first eigenvectors of the normalized kernel Gram matrix $\frac{1}{n}(k(x_i, x_j))_{1 \leq i,j \leq n}$. At this point we can see the relation appear with Kernel PCA. We next precise this relation and give an interpretation of the resulting algorithm in terms of dimensionality reduction.

### 2.3 Link with Kernel Principal Component Analysis

Principal Component Analysis (PCA), and its non-linear variant, KPCA are widely used algorithms in data analysis. They extract from the input data space a basis $(v_i)_{i \geq 1}$ which is, in some sense, adapted to the data by looking for directions where the variance is maximized. They are often used as a pre-processing on the data in order to reduce the dimensionality or to perform de-noising.

As will be made more explicit in the next section, the Kernel Projection Machine consists in replacing the ideal projection estimator defined by (3) by

$$\widehat{f}_D = \underset{f \in S_D}{\operatorname{argmin}} \frac{1}{n} \sum_{i=1}^{n} (1 - y_i f(X_i))_+$$

where $S_D$ is the space of dimension $D$ chosen by the first $D$ principal components chosen by KPCA in feature space. Hence, roughly speaking, in the KPM, the SVM penalization is replaced by dimensionality reduction.

Choosing $D$ amounts to selecting the optimal $D$-dimensional representation of our data for the classification task, in other words to extracting the information that is needed for this task by model selection taking into account the relevance of the directions for the classification task.

To conclude, the KPM is a method of dimensionality reduction that takes into account the labels of the training data to choose the "best" dimension.

## 3 The Kernel Projection Machine Algorithm

In this section, the empirical (and computable) version of the KPM algorithm is derived from the previous theoretical arguments.

In practice the true eigenfunctions of the kernel operator are not computable. But since only the values of functions $\phi_1, \ldots, \phi_D$ at points $x_1, \ldots, x_n$ are needed for minimizing the empirical risk over $\mathcal{F}_D$, the eigenvectors of the kernel matrix $K = (k(x_i, x_j))_{1 \leq i,j \leq n}$ will be enough for our purpose. Indeed, it is well known in numerical analysis (see [2]) that the eigenvectors of the kernel matrix approximate the eigenfunctions of the kernel operator. This result has been pointed out in [7] in a more probabilistic language. More precisely, if $V_1, \ldots, V_D$ denote the $D$ first eigenvectors of $K$ with associated eigenvalues $\widehat{\lambda}_1 \geq \widehat{\lambda}_2 \geq \ldots \geq \widehat{\lambda}_D$, then for each $V_i$

$$V_i = \left( V_i^{(1)}, \ldots, V_i^{(n)} \right) \approx (\phi_i(x_1), \ldots, \phi_i(x_n)) \tag{5}$$

Hence, considering Equation (4), the empirical version of the algorithm described above will first consist of solving, for each dimension $D$, the following optimization problem:

$$(\beta^*, b^*) = \arg \min_{\beta \in \mathbb{R}^D, b \in \mathbb{R}} \sum_{i=1}^{n} \left( 1 - y_i \left( \sum_{j=1}^{D} \beta_j V_j^{(i)} + b \right) \right)_+ \tag{6}$$

Then the solution should be

$$\hat{f}_D(.) = \sum_{j=1}^{D} \beta_j^* \phi_j(.) + b^* . \tag{7}$$

Once again the true functions $\phi_j$'s are unknown. At this stage, we can do an expansion of the solution in terms of the kernel similarly to the SVM algorithm, in the following way:

$$\hat{f}_D(.) = \sum_{i=1}^{n} \alpha_i^* k(x_i, .) + b^* \tag{8}$$

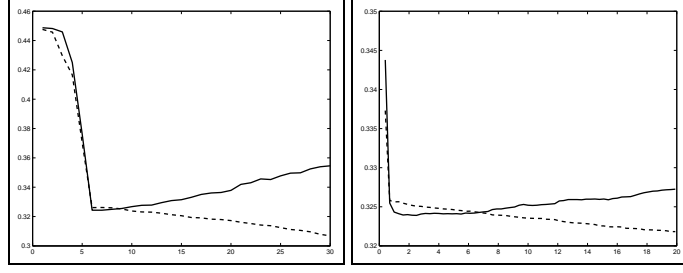

Figure 1: **Left**: KPM risk (solid) and empirical risk (dashed) versus dimension D. **Right**: SVM risk and empirical risk versus C. Both on dataset 'flare-solar'.

Narrowing both expressions ( 7) and ( 8) at points $x_1, \ldots, x_n$ leads the following equation:

$$\beta_1^* V_1 + \ldots + \beta_D^* V_D = K\alpha^* \tag{9}$$

which has a straightforward solution: $\alpha^* = \sum_{j=1}^{D} \frac{\beta_j^*}{\widehat{\lambda}_j} V_j$ (provided the $D$ first eigenvalues are all strictly positive).

Now the KPM algorithm can be summed up as follows:

1. given data $x_1, \ldots, x_n \in \mathcal{X}$ and a positive kernel $k$ defined on $\mathcal{X} \times \mathcal{X}$, compute the kernel matrix $K$ and its eigenvectors $V_1, \ldots, V_n$ together with its eigenvalues in decreasing order $\widehat{\lambda}_1 \geq \widehat{\lambda}_2 \geq \ldots \geq \widehat{\lambda}_n$.

2. for each dimension $D$ such that $\widehat{\lambda}_D > 0$ solve the linear optimization problem

$$(\beta^*, b^*) = \arg\min_{\beta, b, \xi} \sum_{i=1}^{n} \xi_i \tag{10}$$

under constraints $\forall i = 1 \ldots n, \xi_i \geq 0 , y_i \left( \sum_{j=1}^{D} \beta_j V_j^{(i)} + b \right) \geq 1 - \xi_i .$ (11)

Next, compute $\alpha^* = \sum_{j=1}^{D} \frac{\beta_j^*}{\widehat{\lambda}_j} V_j$ and $\hat{f}_D(.) = \sum_{i=1}^{n} \alpha_i^* k(x_i, .) + b^*$

3. The last step is a model selection problem: choose a dimension $\hat{D}$ for which $\hat{f}_{\hat{D}}$ performs well. We do not address directly this point here; one can think of applying cross-validation, or to penalize the empirical loss by a penalty function depending on the dimension.

## 4   Experiments

The KPM was implemented in Matlab using the free library GLPK for solving the linear optimization problem. Since the algorithm involves the eigendecomposition of the kernel matrix, only small datasets have been considered for the moment.

In order to assess the performance of the KPM, we carried out experiments on benchmark datasets available on Gunnar Rätsch's web site [3]. Several state-of-art algorithms have already been applied to those datasets, among which the SVM. All results are reported on the web site. To get a valid comparison with the SVM, on each classification task, we used

Table 1: Test errors of the KPM on several benchmark datasets, compared with SVM, using G.Rätsch's parameter selection procedure (see text). As an indication the best of the six results presented in [3] are also reported.

|  | KPM | (selected $D$) | SVM | Best of 6 |
|---|---|---|---|---|
| Banana | $10.73 \pm 0.42$ | 15 | $11.53 \pm 0.66$ | $10.73 \pm 0.43$ |
| Breast Cancer | $26.51 \pm 4.75$ | 24 | $26.04 \pm 4.74$ | $24.77 \pm 4.63$ |
| Diabetis | $23.37 \pm 1.92$ | 11 | $23.53 \pm 1.73$ | $23.21 \pm 1.63$ |
| Flare Solar | $32.43 \pm 1.85$ | 6 | $32.43 \pm 1.82$ | $32.43 \pm 1.82$ |
| German | $23.59 \pm 2.15$ | 14 | $23.61 \pm 2.07$ | $23.61 \pm 2.07$ |
| Heart | $16.89 \pm 3.53$ | 10 | $15.95 \pm 3.26$ | $15.95 \pm 3.26$ |

Table 2: Test errors of the KPM on several benchmark datasets, compared with SVM, using standard 5-fold cross-validation on each realization.

|  | KPM | SVM |
|---|---|---|
| Banana | $11.14 \pm 0.73$ | $10.69 \pm 0.67$ |
| Breast Cancer | $26.55 \pm 4.43$ | $26.68 \pm 5.23$ |
| Diabetis | $24.14 \pm 1.86$ | $23.79 \pm 2.01$ |
| Flare Solar | $32.70 \pm 1.97$ | $32.62 \pm 1.86$ |
| German | $23.82 \pm 2.23$ | $23.79 \pm 2.12$ |
| Heart | $17.59 \pm 3.30$ | $16.23 \pm 3.18$ |

the same kernel parameters as those used for SVM, so as to work with exactly the same geometry.

There is a subtle, but important point arising here. In the SVM performance reported by G. Rätsch, the regularization parameter $C$ was first determined by cross-validation on the first 5 realizations of each dataset; then the median of these values was taken as a fixed value for the other realizations. This was done apparently for saving computation time, but this might lead to an over-optimistic estimation of the performances since in some sense some extraneous information is then available to the algorithm and the variation due to the choice of $C$ is reduced to almost zero. We first tried to mimic this methodology by applying it, in our case, to the choice of $D$ itself (the median of 5 $D$ values obtained by cross-validation on the first realizations was then used on the other realizations).

One might then argue that this way we are selecting a *parameter* by this method instead of a *meta-parameter* for the SVM, so that the comparison is unfair. However, this distinction being loose, this a rather moot point. To avoid this kind of debate and obtain fair results, we decided to re-run the SVM tests by selecting systematically the regularization parameter by a 5-fold cross-validation on each training set, and for our method, apply the same procedure to select $D$. Note that there is still extraneous information in the choice of the kernel parameters, but at least it is the same for both algorithms.

Results relative to the first methodology are reported in table 1, and those relative the second one are reported in table 2. The globally worst performances exhibited in the second table show that the first procedure may indeed be too optimistic. It is to be mentionned that the parameter $C$ of the SVM was systematically sought on a grid of only 100 values, ranging from 0 to three times the optimal value given in [3]. Hence those experimental results are to be considered as preliminary, and in no way they should be used to establish a significant difference between the performances of the KPM and the SVM. Interestingly, the graphic on the left in Figure 4 shows that our procedure is very different from the one of [8]: when $D$ is very large, our risk increases (leading to the existence of a minimum) while the risk of [8] always decreases with $D$.

# 5  Conclusion and discussion

To summarize, one can see the KPM as an alternative to the regularization of the SVM: regularization using the RKHS norm can be replaced by finite dimensional projection. Moreover, this algorithm performs KPCA towards classification and thus offers a criterion to decide what is the right order of expansion for the KPCA.

Dimensionality reduction can thus be used for classification but it is important to keep in mind that it behaves like a regularizer. Hence, it is clearly useless to plug it in a classification algorithm that is already regularized: the effect of the dimensionality reduction may be canceled as noted by [8].

Our experiments explicitly show the regularizing effect of KPCA: no other smoothness control has been added in our algorithm and still, it gives performances comparable to the one of SVM provided the dimension $D$ is picked correctly. We only considered here selection of $D$ by cross-validation; other methods such as penalization will be studied in future works. Moreover, with this algorithm, we obtain a $D$-dimensional representation of our data which is optimal for the classification task. Thus KPM can be see as a de-noising method who takes into account the labels.

This version of the KPM only considers one kernel and thus one vectorial space by dimension. A more advanced version of this algorithm is to consider several kernels and thus choose among a bigger family of spaces. This family then contains more than one space by dimension and will allow to directly compare the performance of different kernels on a given task, thus improving efficiency for the dimensionality reduction while taking into account the labels.

# References

[1] P. Massart A. Barron, L. Birgé. Risk bounds for model selection via penalization. *Proba.Theory Relat.Fields*, 113:301–413, 1999.

[2] Baker. *The numerical treatment of integral equations*. Oxford:Clarendon Press, 1977.

[3] http://ida.first.gmd.de/~raetsch/data/benchmarks.htm. Benchmark repository used in several Boosting, KFD and SVM papers.

[4] G. Blanchard, O. Bousquet, and P.Massart. Statistical performance of support vector machines. *Manuscript*, 2004.

[5] D.L. Donoho, R.C. Liu, and B. MacGibbon. Minimax risk over hyperrectangles, and implications. *Ann. Statist. 18,1416-1437*, 1990.

[6] T. Evgeniou, M. Pontil, and T. Poggio. Regularization networks and support vector machines. In A. J. Smola, P. L. Bartlett, B. Schölkopf, and D. Schuurmans, editors, *Advances in Large Margin Classifiers*, pages 171–203, Cambridge, MA, 2000. MIT Press.

[7] V. Koltchinskii. Asymptotics of spectral projections of some random matrices approximating integral operators. *Progress in Probability*, 43:191–227, 1998.

[8] B. Schölkopf, A. J. Smola, and K.-R. Müller. Nonlinear component analysis as a kernel eigenvalue problem. *Neural Computation*, 10:1299–1319, 1998.

[9] A. J. Smola and B. Schölkopf. On a kernel-based method for pattern recognition, regression, approximation and operator inversion. *Algorithmica*, 22:211–231, 1998.

[10] G. Wahba. *Spline Models for Observational Data*, volume 59 of *CBMS-NSF Regional Conference Series in Applied Mathematics*. Society for Industrial and Applied Mathematics, Philadelphia, Pennsylvania, 1990.
